# Efficient algorithms for learning kernels from multiple similarity matrices with general convex loss functions

**Achintya Kundu**
Dept. of Computer Science & Automation,
Indian Institute of Science, Bangalore.
`achintya@csa.iisc.ernet.in`

**Vikram Tankasali**
Dept. of Computer Science & Automation,
Indian Institute of Science, Bangalore.
`vikram@csa.iisc.ernet.in`

**Chiranjib Bhattacharyya**
Dept. of Computer Science & Automation,
Indian Institute of Science, Bangalore.
`chiru@csa.iisc.ernet.in`

**Aharon Ben-Tal**
Faculty of Industrial Engg. & Management,
Technion - Israel Institute of Technology, Haifa.
`abental@ie.technion.ac.il`
Visiting Professor, CWI, Amsterdam

## Abstract

In this paper we consider the problem of learning an $n \times n$ kernel matrix from $m(\geq 1)$ similarity matrices under general convex loss. Past research have extensively studied the $m = 1$ case and have derived several algorithms which require sophisticated techniques like ACCP, SOCP, etc. The existing algorithms do not apply if one uses arbitrary losses and often can not handle $m > 1$ case. We present several provably convergent iterative algorithms, where each iteration requires either an SVM or a Multiple Kernel Learning (MKL) solver for $m > 1$ case. One of the major contributions of the paper is to extend the well known Mirror Descent(MD) framework to handle Cartesian product of psd matrices. This novel extension leads to an algorithm, called EMKL, which solves the problem in $O(\frac{m^2 \log n}{\epsilon^2})$ iterations; in each iteration one solves an MKL involving $m$ kernels and $m$ eigen-decomposition of $n \times n$ matrices. By suitably defining a restriction on the objective function, a faster version of EMKL is proposed, called REKL, which avoids the eigen-decomposition. An alternative to both EMKL and REKL is also suggested which requires only an SVM solver. Experimental results on real world protein data set involving several similarity matrices illustrate the efficacy of the proposed algorithms.

## 1 Introduction

Learning procedures based on positive semidefinite (psd) kernel functions, like Support vector machines (SVMs), have emerged as powerful tools for several learning tasks with wide applicability [13]. In many applications it is relatively straightforward to define measures of similarity between any pair of examples but extremely difficult to design a kernel function for accurate classification. For instance, similarity score between two protein sequences given by various measures like BLAST [1], Smith-Waterman [14], etc are not psd, whence cannot be substituted as kernel. In this paper, we consider the problem of learning an optimal kernel matrix, from multiple similarity matrices, that yields accurate classification.

Let the set of $n \times n$ real symmetric matrices be denoted as $\mathbb{S}^n$ and the set of psd matrices as $\mathbb{S}^n_+$. Consider a binary classification problem with $n$ training examples. Let $\mathbf{y} \in \{+1, -1\}^n$ be the

vector of class labels and $K \in \mathbb{S}_+^n$ be a kernel matrix. The SVM formulation [13] computes a performance measure $\omega(K)$ by solving

$$\omega(K) = \max_{\boldsymbol{\alpha} \in \mathcal{A}} \left[ \boldsymbol{\alpha}^\top \mathbf{1} - 0.5\, \boldsymbol{\alpha}^\top Y K Y \boldsymbol{\alpha} \right], \tag{1}$$

where $\mathcal{A} = \{ \boldsymbol{\alpha} \in \mathbb{R}^n \mid \boldsymbol{\alpha}^\top \mathbf{y} = 0, \, 0 \leq \boldsymbol{\alpha} \leq C\mathbf{1} \}$, $Y = \mathrm{diag}(\mathbf{y})$, $\mathbf{1} = [1 \ldots 1]^\top \in \mathbb{R}^n$ and $C$ user defined positive constant.

## 1.1 Background and Related work

To the best of our knowledge the problem of handling multiple similarity matrices and arbitrary convex losses has not been studied before. Existing literature has focussed on only one similarity matrix and specific choices of loss function. In this section we briefly review the related literature.

The problem was first studied in [8] for a single similarity matrix. They introduced the following optimization problem

$$\min_{K \in \mathbb{S}_+^n} \omega(K) + \rho \|K - S\|_F^2, \tag{2}$$

where $S \in \mathbb{S}^n$, is a similarity matrix, whose $(i,j)$-th element $S(i,j)$ represents the similarity between example pair $i, j$ and $\omega(K)$ is defined in (1). By interchanging the maximization over $\boldsymbol{\alpha}$ and minimization over $K$ the authors note that the inner minimization admits a closed form solution:

$$K^* = \left( S + (4\rho)^{-1}(Y\boldsymbol{\alpha})(Y\boldsymbol{\alpha})^\top \right)_+, \tag{3}$$

where $(X)_+$ denotes the psd matrix obtained by clipping the negative eigen values of $X$ to zero, i.e., if $X = \sum_{i=1}^n \lambda_i v_i v_i^\top$ is the eigen decomposition of $X \in \mathbb{S}^n$ then $(X)_+ = \sum_{i=1}^n \max(\lambda_i, 0) v_i v_i^\top$. After plugging in the value of $K^*$ authors suggest using sophisticated techniques like Analytic center cutting plane (ACCP) method to solve the outer maximization in $\boldsymbol{\alpha}$.

The formulation (2) was studied further in [5] where an iterative algorithm based on solving a quadratically constrained linear program (QCLP) was proposed. In both the above approaches the order of maximization and minimization has been interchanged which lead to optimization problems that can be posed as semi-infinite quadratically constrained linear Programs (SIQCLP) [5]. In [6] an alternate loss function $\|K - S\|_F$ was studied which led to a Second Order Cone Program(SOCP) formulation. The choice of $\|K - S\|_F^2$, as a measure of loss, is arbitrary. In this paper we generalize the setting in (2) by providing an algorithm which works for any convex loss function. We note that the method used in solving (2) utilizing (3) is specific to the loss function $\|K - S\|_F^2$ and do not apply generally. Apart from non-applicability of the existing methods to other loss functions it is not clear how these procedures could be used to handle multiple similarity matrices.

**Contributions:** The key contribution of the paper is to design efficient procedures which can learn a kernel matrix, $K \in \mathbb{S}_+^n$, from $m(\geq 1)$ similarity matrices, possibly indefinite, under general convex sub-differentiable loss function by using either SVM or MKL solvers. We study the problem in two different settings. In the first setting we consider learning a single kernel matrix from multiple similarity matrices under a general loss function. A novel algorithm, referred in the paper as ESKL, based on the Mirror Descent (MD) [3] procedure is proposed. It is a provably convergent algorithm which requires $O(\frac{\log n}{\epsilon^2})$ calls to an SVM solver. In the second setting we consider learning separate kernel matrix for each of the given similarity matrices and then aggregating them using a Multiple Kernel Learning (MKL) setup. The resultant formulation is non-trivial to solve. We present EMKL which is based on generalizing the existing MD setup to deal with Cartesian product of psd matrices. Like the previous case it requires $O(\frac{m^2 \log n}{\epsilon^2})$ calls to an MKL solver. At every iteration the algorithm also requires eigen-decomposition which is expensive. We present a related algorithm, REKL, which does not require the expensive eigen-decomposition step but yields similar classification performance as EMKL. Apart from allowing general loss functions the procedures also opens up new avenues for learning multiple kernels, which could be viable alternatives to the framework proposed in [7].

The remainder of the paper is organized as follows: in section 2 we discuss problem formulation. Our main contribution is in section 3, where we develop mirror descent algorithms for learning kernel from multiple indefinite similarity matrices. We also analyze the complexity and convergence properties of the proposed algorithms. Finally we present our experimental results in section 4.

## 2  Problem formulation

Given multiple similarity matrices $\{S_i \; : \; i = 1, \ldots, m\}$ we consider the following formulation

$$\min_{K \in \mathbb{S}_+^n, \, \mathrm{tr}(K) = \tau} \quad f(K) \;\equiv\; \Big[ \; \omega(K) \;+\; \rho \sum_{i=1}^m L_i\,(K - S_i) \; \Big], \tag{4}$$

where $\rho \geq 0$ is a trade-off parameter, $L_i : \mathbb{S}^n \to \mathbb{R}$ is a convex sub-differentiable loss function operating on $K$ and $S_i$. We impose the trace constraint on $K$ to ensure good generalization as in [7].

A more naturally suited formulation for handling multiple similarity matrices is to consider learning individual kernel matrix $K_i$ from similarity matrix $S_i$, $\forall i$ and invoke a Multiple Kernel Learning (MKL) setup to obtain a kernel matrix $K \in \mathbb{K} \triangleq \big\{ \sum_i \beta_i\,K_i \mid K_i \in \mathbb{S}_+^n, \; \beta_i \geq 0, \; \forall i \big\}$. In [7] the MKL problem is proposed as

$$\Omega(K_1, \ldots, K_m) = \min_{K \in \mathbb{K}, \, \mathrm{tr}(K) = \tau} \omega(K)\,, \tag{5}$$

where the kernels $K_i$'s are fixed and $\beta_i$'s are variable. Based on MKL we consider the following kernel learning formulation

$$\min_{K_1, \ldots, K_m} \quad F(K_1, \ldots, K_m) \;\equiv\; \Big[ \; \Omega(K_1, \ldots, K_m) \;+\; \rho \sum_{i=1}^m L_i(K_i - S_i) \; \Big],$$
$$s.t. \quad K_i \in \mathbb{S}_+^n\,, \;\; \mathrm{tr}(K_i) = \tau\,, \;\; i = 1, \ldots, m. \tag{6}$$

Note that $\Omega(K_1, \ldots, K_m)$ can be obtained by solving any standard MKL formulation.

The restriction of $\Omega(K_1, \ldots, K_m)$ on the Cartesian product of sets $\bigotimes_{i=1}^m \{ K_i = \sum_j \mu_{ij}\,\mathbf{v}_{ij}\,\mathbf{v}_{ij}^\top \mid \mu_{ij} \geq 0, \; \mathbf{v}_{ij} \text{ is } j\text{-th eigen vector of } S_i \}$, yields a very interesting alternative to (6). Based on this restriction we formulate the restricted kernel learning problem as

$$\min_{\boldsymbol{\mu}_1, \ldots, \boldsymbol{\mu}_m \in \mathbb{R}^n} \quad g(\boldsymbol{\mu}_1, \ldots, \boldsymbol{\mu}_m) \;\equiv\; \Big[ \; \Omega(K_1, \ldots K_m) \;+\; \rho \sum_{i=1}^m \ell_i(\boldsymbol{\mu}_i, \boldsymbol{\lambda}_i) \; \Big],$$
$$s.t. \quad K_i = \sum_j \mu_{ij}\,\mathbf{v}_{ij}\,\mathbf{v}_{ij}^\top, \;\; K_i \in \mathbb{S}_+^n, \;\; \sum_{j=1}^m \mu_{ij} = \tau, \;\; i = 1, \ldots, m, \tag{7}$$

where $\boldsymbol{\lambda}_i = [\lambda_{i1} \ldots \lambda_{in}]^\top$ denotes the eigen values of $S_i$ and $\ell_i : \mathbb{R}^n \times \mathbb{R}^n \to \mathbb{R}$ is a convex loss function on $\boldsymbol{\mu}_i = [\mu_{i1} \ldots \mu_{in}]^\top$.

We mention that the formulation (4) generalizes the existing single similarity matrix based formulations. For $m = 1$ with $L(X) = \|X\|_F^2$, $L(X) = \|X\|_F$ we recover the formulations in [8] and [6] respectively (albeit with a trace constraint). Also the SOCP based spectrum modification learning formulation [6] proposed in the context of single similarity matrix ($m = 1$) is a special case of (7).

## 3  Kernel Learning using Mirror Descent

In this section we derive general methods for solving (4) and (6) based on the following assumptions: loss function $L_i$ is convex; a sub-gradient $L_i'$ can be computed efficiently and the computed sub-gradients are bounded. We also assume the availability of an efficient SVM / MKL solver.

### 3.1  Entropic single kernel learning (ESKL) algorithm

We denote the feasible set of kernels as $\mathcal{K} = \{ K \in \mathbb{S}_+^n \mid \mathrm{tr}(K) = \tau \}$ and its relative interior as $\mathrm{int}(\mathcal{K}) = \{ K \in \mathbb{S}^n \mid \mathrm{tr}(K) = \tau, \; K \text{ is positive definite } \}$. Note that $\mathcal{K}$ is convex and compact. Define inner-product on $\mathbb{S}^n$ as $\langle K, K' \rangle = \mathrm{tr}(KK')$. From Eqn. (1) we note that $\omega(K)$ is a convex function of $K$. Therefore the objective function $f$ in (4) is convex and Lipschitz continuous on $\mathcal{K}$. Let $\boldsymbol{\alpha}^*$ denote a maximizer of the SVM dual (1). Then we can compute a sub-gradient of $f$ as

$$f'(K) \;=\; -0.5\,Y\,\boldsymbol{\alpha}^*\boldsymbol{\alpha}^{*\top}\,Y \;+\; \rho \sum_{i=1}^m L_i'\,(K - S_i)\,. \tag{8}$$

Thus the convex programming problem (4) satisfies the conditions for applying Mirror Descent (MD) [2] scheme. To apply MD procedure we require a strongly convex and continuously differentiable function $\psi : \mathcal{K} \to \mathbb{R}$. Following [2] we choose negative of matrix entropy as the candidate for $\psi$:

$$\psi(K) \;=\; \sum_{j=1}^n \lambda_j \, \log \lambda_j\,, \tag{9}$$

where $(\lambda_1, \ldots, \lambda_n)$ are the eigen values of $K \in \mathcal{K}$. With the above setup we derive an MD algorithm, named entropic single kernel learning (ESKL) algorithm, similar to the entropic mirror descent algorithm proposed in [2].

---

**Algorithm 1** Entropic single kernel learning (ESKL) algorithm

---

Initialization: $K^{(1)} \in \text{int}(\mathcal{K})$. Set $t = 0$.
**repeat**
- $t := t + 1$.
- Obtain $\boldsymbol{\alpha}^*$ i.e. a maximizer of the SVM dual problem (1) for kernel $K = K^{(t)}$.
- Compute sub-gradient $f'(K^{(t)}) := -0.5 \, Y \boldsymbol{\alpha}^* \boldsymbol{\alpha}^{*\top} Y \, + \, \rho \sum_j L_i'(K^{(t)} - S_i)$.
- Choose suitable step-size $\eta_t$.
- Compute eigen decomposition $f'(K^{(t)}) = V^{(t)} \, \text{diag}([d_1^{(t)} \, \ldots \, d_n^{(t)}]) \, V^{(t)\top}$.
- $\lambda_j^{(t+1)} := \dfrac{\tau \, \lambda_j^{(t)} \, \exp(-\eta_t \, d_j^{(t)})}{\sum_{l=1}^n \lambda_l^{(t)} \, \exp(-\eta_t \, d_l^{(t)})}, \; \forall j = 1, \ldots, n$.
- $K^{(t+1)} := V^{(t)} \, \text{diag}\left( \left[ \lambda_1^{(t+1)} \, \ldots \, \lambda_n^{(t+1)} \right] \right) \, V^{(t)\top}$.

**until** Convergence

---

**Proposition 3.1.** *Let $f^{(t)}$ denote the objective function value at $t$-th iteration and $f^*$ be the optimal objective value. If the ESKL algorithm is initialized with $K^{(1)} = \frac{\tau}{n} I$ and the step-sizes are chosen as* $\quad \eta_t = \dfrac{1}{\text{Lip}(f)} \sqrt{\dfrac{2 \log n}{t}} \quad$ *then* $\quad \min\limits_{1 \leq t \leq T} f^{(t)} - f^* \; \leq \; \tau \, \text{Lip}(f) \sqrt{\dfrac{2 \log n}{T}},$ *where* $\text{Lip}(f)$ *is a Lipschitz constant of $f$ such that* $\| f'(K^{(t)}) \|_F \; \leq \; \text{Lip}(F) , \; t = 1, \ldots, T$.*

*Proof.* The strong convexity constant of $\psi$ w.r.t. $\| \cdot \|_F$ norm is $\sigma = \frac{1}{\tau}$. Let $\text{B}_\psi$ denote the Bregman distance function [2] generated by $\psi$. Then we have $\text{B}_\psi(K, K^{(1)}) \leq \tau \log n , \; \forall K \in \mathcal{K}$ (assuming $n \geq 3$). We complete the proof by applying Theorem 4.2 of [2] to the ESKL algorithm. $\qquad \square$

### 3.2 Entropic multiple kernel learning (EMKL) algorithm

Consider the kernel learning formulation (6) which minimizes the distances of kernels $\{K_i \, : \, i = 1, \ldots, m\}$ from the corresponding similarity matrices $\{S_i \, : \, i = 1, \ldots, m\}$ and simultaneously learns an SVM classifier by performing multiple kernel learning (MKL) on those kernels. To learn a non-sparse combination of kernels the following MKL formulation has been proposed in [10]:

$$\Omega(K_1, \ldots, K_m) \; \equiv \; \max_{\boldsymbol{\gamma} \in \triangle_m} \; \max_{\boldsymbol{\alpha} \in \mathcal{A}} \; \left[ \, \boldsymbol{\alpha}^\top \mathbf{1} - \frac{1}{2} \boldsymbol{\alpha}^\top Y \Big( \sum_{i=1}^m \frac{1}{\gamma_i} K_i \Big) Y \boldsymbol{\alpha} \, \right], \tag{10}$$

where $\triangle_m = \left\{ \boldsymbol{\gamma} = [\gamma_1 \, \ldots \, \gamma_m]^\top : \sum_i \gamma_i \leq 1, \, \gamma_i \geq 0, \forall i \right\}$. With $\Omega(K_1, \ldots, K_m)$ as defined above, the objective function $F$ in (6) can be expressed as

$$F(K_1, \ldots, K_m) = \max_{\boldsymbol{\gamma} \in \triangle_m} \; \max_{\boldsymbol{\alpha} \in \mathcal{A}} \; \sum_i F_i(K_i \, ; \, \boldsymbol{\alpha}, \boldsymbol{\gamma}),$$

$$F_i(K_i \, ; \, \boldsymbol{\alpha}, \boldsymbol{\gamma}) \; = \; \tfrac{1}{m} \mathbf{1}^\top \boldsymbol{\alpha} - \tfrac{1}{2 \gamma_i} \text{tr}\left( K_i Y \boldsymbol{\alpha} \boldsymbol{\alpha}^\top Y \right) \, + \, \rho \, L_i(K_i - S_i), \; i = 1, \ldots, m. \tag{11}$$

Let $\mathbb{V} := \bigotimes_{i=1}^m \mathbb{S}^n$, $\mathcal{K} = \{K \in \mathbb{S}_+^n \, : \, \text{tr}(K) = \tau\}$ and $\mathcal{K}^m := \bigotimes_{i=1}^m \mathcal{K} \subset \mathbb{V}$. Denote $\mathbf{K} = (K_1, \ldots, K_m) \in \mathcal{K}^m$. Define inner product on $\mathbb{V}$ as $\langle \mathbf{K}, \mathbf{K}' \rangle_\mathbb{V} := \sum_{i=1}^m \langle K_i, K_i' \rangle$, where $\langle K_i, K_i' \rangle = \text{tr}(K_i K_i')$. Also define a norm on $\mathbb{V}$ as $\|\mathbf{K}\| = \sum_{i=1}^m \|K_i\|_F$. From (10) we note that $\Omega(K_1, \ldots, K_m)$ is a convex function of $(K_1, \ldots, K_m)$ over the compact space $\mathcal{K}^m$. Thus the objective function $F$ in (6) is convex and Lipschitz continuous on $\mathcal{K}^m$.

**Lemma 3.1.** *Let $(\boldsymbol{\alpha}^*, \boldsymbol{\gamma}^*)$ be a solution of (10) and $L_i'$ be a sub-gradient of $L_i$. Then a sub-gradient of $F$ is given by*

$$F'(K_1, \ldots, K_m) \; = \; \left( \, \partial_{K_1} F_1(K_1; \boldsymbol{\alpha}^*, \boldsymbol{\gamma}^*) \; \cdots \; \partial_{K_m} F_m(K_m; \boldsymbol{\alpha}^*, \boldsymbol{\gamma}^*) \, \right),$$

$$\partial_{K_i} F_i(K_i; \boldsymbol{\alpha}^*, \boldsymbol{\gamma}^*) \; = \; -\frac{1}{2 \, \gamma_i^*} Y \boldsymbol{\alpha}^* \boldsymbol{\alpha}^{*\top} Y \, + \, L_i'(K_i - S_i), \; i = 1, \ldots, m. \tag{12}$$

*Proof.* First, we observe that $F_i(K_i; \boldsymbol{\alpha}, \boldsymbol{\gamma})$ is a convex function of $K_i \in \mathcal{K}$ and the expression of $\partial_{K_i} F_i(K_i; \boldsymbol{\alpha}, \boldsymbol{\gamma})$ given in Eqn. (12) is precisely a sub-gradient of $F_i$. Therefore, we can write

$$F_i(K_i'; \boldsymbol{\alpha}, \boldsymbol{\gamma}) \geq F_i(K_i; \boldsymbol{\alpha}, \boldsymbol{\gamma}) + \langle K_i' - K_i, \partial_{K_i} F_i(K_i; \boldsymbol{\alpha}, \boldsymbol{\gamma}) \rangle, \quad \forall K_i' \in \mathcal{K}. \tag{13}$$

By optimality of $(\boldsymbol{\alpha}^*, \boldsymbol{\gamma}^*)$ we have $F(K_1, \ldots, K_m) = \sum_{i=1}^{m} F_i(K_i; \boldsymbol{\alpha}^*, \boldsymbol{\gamma}^*)$. Because of the max operation over $\boldsymbol{\alpha}, \boldsymbol{\gamma}$, we have $F(K_1', \ldots, K_m') \geq \sum_{i=1}^{m} F_i(K_i'; \boldsymbol{\alpha}^*, \boldsymbol{\gamma}^*)$ for any $\mathbf{K}' = (K_1', \ldots, K_m') \in \mathcal{K}^m$. Applying (13) we arrive at

$$F(K_1', \ldots, K_m') \geq F(K_1, \ldots, K_m) + \left\langle \mathbf{K}' - \mathbf{K}, F'(K_1, \ldots, K_m) \right\rangle_{\mathbb{V}},$$

Hence, $F'(K_1, \ldots, K_m)$ given in (12) is a sub-gradient of $F$. $\qquad\square$

We develop a novel Mirror Descent procedure for problem (6) by defining a strongly convex and continuously differentiable function $\Psi$ on the product space $\mathcal{K}^m$ as

$$\Psi(\mathbf{K}) = \sum_{i=1}^{m} \sum_{j=1}^{n} \lambda_{i,j} \log \lambda_{i,j}, \quad \mathbf{K} \in \mathcal{K}^m, \tag{14}$$

where $(\lambda_{i,1}, \ldots, \lambda_{i,n})$ denote eigen values of $K_i$. The resulting algorithm, named entropic multiple kernel learning (EMKL), is given below.

---

**Algorithm 2** Entropic multiple kernel learning (EMKL) algorithm

---

Initialization: $K_i^{(1)} \in \text{int}(\mathcal{K})$, $i = 1, \ldots, m$. Set $t = 0$.
**repeat**
$\quad t := t + 1$.
$\quad$ Obtain $\boldsymbol{\alpha}^*, \boldsymbol{\gamma}^*$ by solving the MKL problem (10) with $K_i = K_i^{(t)}$, $i = 1, \ldots, m$.
$\quad$**for** $i = 1$ **to** $m$ **do**
$\qquad \bullet$ Compute sub-gradient $\partial_{K_i} F_i(K_i^{(t)}; \boldsymbol{\alpha}^*, \boldsymbol{\gamma}^*) := -\frac{1}{2\gamma_i^*} Y \boldsymbol{\alpha}^* \boldsymbol{\alpha}^{*\top} Y + L_i'(K_i^{(t)} - S_i)$.
$\qquad \bullet$ Find eigen decomposition $\partial_{K_i} F_i(K_i^{(t)}; \boldsymbol{\alpha}^*, \boldsymbol{\gamma}^*) = V_i^{(t)} \text{diag}([d_{i,1}^{(t)} \ldots d_{i,n}^{(t)}]) V_i^{(t)\top}$.
$\qquad \bullet$ $\lambda_{i,j}^{(t+1)} := \dfrac{\tau \lambda_{i,j}^{(t)} \exp(-\eta_t d_{i,j}^{(t)})}{\sum_{l=1}^{n} \lambda_{i,l}^{(t)} \exp(-\eta_t d_{i,l}^{(t)})}$, $j = 1, \ldots, n$.
$\qquad \bullet$ $K_i^{(t+1)} := V_i^{(t)} \text{diag}([\lambda_{i,1}^{(t+1)} \ldots \lambda_{i,n}^{(t+1)}]) V_i^{(t)\top}$.
$\quad$**end for**
**until** Convergence

---

**Theorem 3.2.** *Let $F^{(t)}$ denote the objective function value at $t$-th iteration and $F^*$ be the optimal objective value. If the EMKL algorithm is initialized with $K_i^{(1)} = \frac{\tau}{n} I$, $\forall i$ and the step-sizes are chosen as $\eta_t = \dfrac{1}{\text{Lip}(F)} \sqrt{\dfrac{2 \log n}{m\,t}}$ then $\min_{1 \leq t \leq T} F^{(t)} - F^* \leq \tau\, m\, \text{Lip}(F) \sqrt{\dfrac{2 \log n}{T}}$, where $\text{Lip}(F)$ is a Lipschitz constant of $F$ such that $\| \partial_{K_i} F(K_i^{(t)}; \boldsymbol{\alpha}, \boldsymbol{\gamma}) \|_F \leq \text{Lip}(F)$, $\forall i, t$.*

*Proof.* Let $\mathbf{K}^* = (K_1^*, \ldots, K_m^*)$ be an optimal solution of (6). Denote $\mathbf{K}^{(t)} = \left( K_1^{(t)}, \ldots, K_m^{(t)} \right)$. We apply the convergence result presented as Theorem 4.2 in [2]. This leads to the following:

$$\eta_t = \frac{1}{\text{Lip}(F)} \sqrt{\frac{2\,\sigma\, \mathsf{B}_\Psi(\mathbf{K}^*, \mathbf{K}^{(t)})}{t}} \Rightarrow \min_{1 \leq t \leq T} F^{(t)} - F^* \leq \text{Lip}(F) \sqrt{\frac{2\, \mathsf{B}_\Psi(\mathbf{K}^*, \mathbf{K}^{(t)})}{\sigma\, T}}, \tag{15}$$

where $\sigma > 0$ is the strong convexity constant of $\Psi$ and $\mathsf{B}_\Psi$ is the Bregman distance function generated by $\Psi$. For the $\Psi$ function defined in Eqn. (14), we have $\sigma = \frac{1}{m\tau}$. Assuming $n \geq 3$, we also have $\mathsf{B}_\Psi(\mathbf{K}, \mathbf{K}^{(1)}) \leq m\,\tau \log n$, $\forall \mathbf{K} \in \mathcal{K}^m$. Substituting values for $\mathsf{B}_\Psi$ and $\sigma$ in (15) we obtain the desired result. $\qquad\square$

### 3.3 Restricted entropic kernel learning (REKL) algorithm

The proposed EMKL algorithm is computationally expensive as it computes eign decomposition of $m$ matrices of dimension $n \times n$ at every iteration. Here we propose an efficient algorithm by considering the restricted kernel learning formulation (7) where $\Omega(K_1, \ldots, K_m)$ is given in Eqn. (10). We denote the feasible set for $\boldsymbol{\mu}_i$ as $\mathcal{X} := \{\boldsymbol{\mu}_i \in \mathbb{R}^n \mid \mu_{ij} \geq 0, \forall j, \sum_{j=1}^n \mu_{ij} = \tau\}$, which is a convex compact subset of $\mathbb{R}^n$. We note that $\Omega$ in (10) when viewed as a function of $\boldsymbol{\mu}_i$'s, is a convex function on the Cartesian product space $\mathcal{X}^m := \bigotimes_{i=1}^m \mathcal{X}$. The loss function $\ell_i$ is assumed to be a convex function of $\boldsymbol{\mu}_i$ with bounded sub-gradients. Hence, the objective function $g$ in (7) is convex and Lipschitz continuous over the compact space $\mathcal{X}^m$. Denote a sub-gradient of $\ell_i$ as $\left[ \partial_{\mu_{i1}} \ell_i(\boldsymbol{\mu}_i, \boldsymbol{\lambda}_i), \ldots, \partial_{\mu_{in}} \ell_i(\boldsymbol{\mu}_i, \boldsymbol{\lambda}_i) \right]^\top$. We can compute a sub-gradient of $\Omega$ as $\Omega' = (\partial_{\mu_{11}}\Omega, \partial_{\mu_{12}}\Omega, \ldots, \partial_{\mu_{nn}}\Omega)$, where $\partial_{\mu_{ij}}\Omega = -\frac{1}{2\gamma_i^*} \boldsymbol{\alpha}^{*\top} Y \mathbf{v}_{ij} \mathbf{v}_{ij}^\top Y \boldsymbol{\alpha}^*$. We derive an MD algorithm, named restricted entropic kernel learning (REKL), by extending the entropic mirror descent scheme [2] to deal with Cartesian product of simplices. This is achieved by defining a strongly convex and continuously differentiable function $\psi_e : \mathcal{X}^m \to \mathbb{R}$ as

$$\psi_e(\boldsymbol{\mu}_1, \ldots, \boldsymbol{\mu}_m) = \sum_{i=1}^m \sum_{j=1}^n \mu_{ij} \log \mu_{ij}, \; \boldsymbol{\mu}_i \in \mathcal{X}, \forall i. \tag{16}$$

---

**Algorithm 3** Restricted entropic kernel learning (REKL) algorithm

Find eigen decomposition: $S_i = \sum_j \lambda_{ij} \mathbf{v}_{ij} \mathbf{v}_{ij}^\top, \; i = 1, \ldots, m$.
Initialization: $\boldsymbol{\mu}_i^{(1)} \in \text{int}(\mathcal{X}), \; i = 1, \ldots, m$. Set $t = 0$.
**repeat**
  $t = t + 1$
  Obtain $\boldsymbol{\alpha}^*, \gamma^*$ by solving the MKL problem (10) with $K_i = \sum_j \mu_{ij}^{(t)} \mathbf{v}_{ij} \mathbf{v}_{ij}^\top, \; i = 1, \ldots, m$.
  **for** $i = 1$ **to** $m$ **do**
   • Compute sub-gradient $g'^{(t)}_{ij} := -\frac{1}{2\gamma_i^*} \boldsymbol{\alpha}^{*\top} Y \mathbf{v}_{ij} \mathbf{v}_{ij}^\top Y \boldsymbol{\alpha}^* + \partial_{\mu_{ij}} \ell_i(\boldsymbol{\mu}_i^{(t)}, \boldsymbol{\lambda}_i)$.
   • $\mu_{ij}^{(t+1)} := \dfrac{\tau \, \mu_{ij}^{(t)} \, \exp\left( -\eta_t \, g'^{(t)}_{ij} \right)}{\sum_{l=1}^n \mu_{il}^{(t)} \, \exp\left( -\eta_t \, g'^{(t)}_{il} \right)}, \; j = 1, \ldots, n$.
  **end for**
**until** Convergence

---

**Proposition 3.2.** *Let $g^{(t)}$ denote the objective function value at $t$-th iteration and $g^*$ be the optimal objective value. If the REKL algorithm is initialized with $\boldsymbol{\mu}_i^{(1)} = \frac{\tau}{n}\mathbf{1}, \forall i$ and the step-sizes are chosen as $\eta_t = \dfrac{1}{\text{Lip}(g)} \sqrt{\dfrac{2\log n}{m\, t}}$ then $\min_{1 \leq t \leq T} g^{(t)} - g^* \leq \tau m \, \text{Lip}(g) \sqrt{\dfrac{2\log n}{T}}$, where $\text{Lip}(g)$ is a Lipschitz constant of $g$ such that $|g'^{(t)}_{ij}| \leq \text{Lip}(g), \; i, = 1, \ldots, m, \; j = 1, \ldots, n, \; t = 1, \ldots, T$.*

*Proof.* The proof is similar to that of Theorem 3.2. □

### 3.4 Discussion

The ESKL formulation requires $O\left(\frac{\log n}{\epsilon^2}\right)$ iterations (see Proposition 3.1), where in each iteration one solves an SVM and eigen-decomposition of $n \times n$ matrix. Both EMKL and REKL formulations require $O\left(\frac{m^2 \log n}{\epsilon^2}\right)$ iterations (see Theorem 3.2 and Proposition 3.2), and in each iteration one solves an MKL problem. However EMKL is more computationally expensive than REKL.

## 4 Experiments and Results

In this section we experimentally compare the proposed kernel learning formulations against IndSVM [8] and the eigen transformation methods: Denoise, Flip, Shift [15]. Given an indefinite similarity matrix $S$ with eigen-decomposition $S = \sum_j \lambda_j \mathbf{v}_j \mathbf{v}_j^\top$, eigen transformation methods generate kernel matrix as $K := \sum_j \mu_j \mathbf{v}_j \mathbf{v}_j^\top$, where choice $\mu_j$'s are: (a) *Denoise:* $\mu_j = \max(\lambda_j, 0)$,

(b) *Flip:* $\mu_j = |\lambda_j|$, (c) *Shift:* $\mu_j = \lambda_j - \delta$, where $\delta = \min\{\lambda_1, \ldots, \lambda_n, 0\}$. We consider the following choices for the loss functions in ESKL / EMKL: $[\mathcal{L}_1]$ $L(K-S) = \sum_{i,j} |K(i,j) - S(i,j)|$, $[\mathcal{L}_2]$ $L(K-S) = \|K-S\|_F$, $[\mathcal{L}_3]$ $L(K-S) = \sum_{i,j} |K(i,j) - S(i,j)|^2$. For REKL we choose $\ell(\boldsymbol{\mu}_j, \boldsymbol{\lambda}_j) = \|\boldsymbol{\mu}_j - \boldsymbol{\lambda}_j\|_2$, i.e., the Euclidean distance. Algorithm parameters are tuned using standard 5 fold cross validation procedure. LibSVM [4] is used as the SVM solver. For each data set we have considered equal number of positive and negative training / test samples. We report classification performance in terms of accuracy and F-score (expressed as % ) averaged over 5 different train / test splits.

## 4.1   Data sets

We experimented on 10 different data sets including the data sets covered in [16, 6]. We have generated the indefinite similarity matrices as prescribed in [16] for each of the following data sets: *Sonar, Liver disorder, Ionosphere, Diabetes* and *Heart*. We have used the same similarity matrices as in [6]:[1] for the data sets *Amazon, AuralSonar, Yeast-SW-5-7* and *Yeast-SW-5-12* .

To test the proposed multiple similarity based formulations we experimented on a subset of the SCOP database [9] taken from Protein Classification Benchmark Collection [2]. Considering proteins having $< 40\%$ sequence identity, we randomly select 8 super-families which have at least 45 proteins. We compute 3 different pairwise similarity measure for proteins: Psi-BLAST [1], Smith-Waterman [14] and Needleman-Wunsch [11]. The similarity matrices obtained from these 3 similarity measures are indefinite in general.

## 4.2   Effect of various loss functions

We experimentally demonstrate the ability of the proposed ESKL algorithm in handling general convex loss function. Classification performance is presented in Table 1. We observe that on *Liver disorder* data set $\mathcal{L}_2$ loss performs better than $\mathcal{L}_1$, $\mathcal{L}_3$. Again, on *Diabetes* and *Heart* data sets both $\mathcal{L}_1$, $\mathcal{L}_2$ provides much better performance better than $\mathcal{L}_3$. From Table 2 we observe that on *AuralSonar* data set ESKL formulation works best with $\mathcal{L}_3$ loss function. But on *Yeast-SW-5-7* data set $\mathcal{L}_1$ loss function works best. Therefore we can say that the choice of loss function has an effect on classification accuracy. This suggest the need for a general algorithm which provides flexibility to choose loss function based on the data set. Hence in this paper we have developed the algorithms keeping the choice of loss function almost open.

Table 1: Comparison of classification accuracy (odd rows) and F-score (even rows) on UCI data sets

| Dataset | Eigen Transformation | | | IndSVM | ESKL | | |
|---|---|---|---|---|---|---|---|
| | Denoise | Flip | Shift | [8] | $[\mathcal{L}_1]$ | $[\mathcal{L}_2]$ | $[\mathcal{L}_3]$ |
| *Sonar* | 71.5 | 72.5 | 76.5 | 76.5 | 75.5 | 73.0 | 75.5 |
| | 70.0 | 70.6 | 75.0 | 74.8 | 73.9 | 71.3 | 74.1 |
| *Liver disorder* | 57.6 | 54.5 | 55.5 | 59.7 | 61.0 | **62.8** | 60.7 |
| | 55.4 | 53.8 | 52.9 | 55.8 | 58.9 | **59.1** | 57.5 |
| *Ionosphere* | 87.3 | 89.6 | 91.2 | 91.5 | 88.5 | 91.2 | 91.5 |
| | 87.6 | 89.9 | 91.4 | 91.8 | 88.5 | 91.4 | 91.8 |
| *Diabetes* | 63.9 | 58.7 | 64.4 | 70.2 | **73.3** | **73.5** | 69.8 |
| | 65.0 | 58.8 | 65.1 | 71.4 | **74.3** | **74.6** | 71.0 |
| *Heart* | 73.3 | 63.8 | 75.8 | 76.3 | **78.8** | **78.8** | 76.3 |
| | 73.1 | 65.1 | 76.9 | 76.5 | **79.5** | **79.0** | 76.5 |

## 4.3   Combining multiple sequence similarity matrices for Proteins

Consider the task of classifying proteins into super-families when multiple sequence similarity measures are available. We perform 1 vs rest classification experiments on each of the 8 protein super-families and report performance averaged over 5 train / test splits. One can extend IndSVM [8]

Table 2: Comparison of classification accuracy (odd rows) and F-score (even rows) on real data sets

| Dataset | Eigen Transformation | | | IndSVM | ESKL | | |
|---|---|---|---|---|---|---|---|
| | Denoise | Flip | Shift | [8] | $[\mathcal{L}_1]$ | $[\mathcal{L}_2]$ | $[\mathcal{L}_3]$ |
| *Amazon* | 83.8 | 83.8 | 85.0 | 87.5 | **88.8** | 85.0 | **88.8** |
| | 84.8 | 84.8 | 85.9 | 86.9 | **88.0** | 84.3 | **88.0** |
| *AuralSonar* | 87.0 | 87.0 | 87.0 | 88.0 | 88.0 | 87.0 | **90.0** |
| | 86.5 | 86.3 | 86.3 | 87.3 | 87.3 | 86.3 | **89.1** |
| *Yeast-SW-5-7* | 75.5 | 70.0 | 74.0 | 77.0 | **79.0** | 75.5 | 76.5 |
| | 77.1 | 72.4 | 74.1 | 77.7 | **79.9** | 76.1 | 77.2 |
| *Yeast-SW-5-12* | 86.0 | 85.5 | 86.0 | 90.0 | 90.0 | 90.5 | 90.0 |
| | 87.1 | 85.8 | 87.6 | 90.9 | 90.9 | 91.35 | 90.9 |

Table 3: Comparison of classification accuracy (odd rows) and F-score (even rows) on Proteins

| Super family | Linear SVM | Eigen Denoise | IndSVM [8] | simple MKL [12] | ESKL $[\mathcal{L}_1]$ | EMKL $[\mathcal{L}_1]$ | REKL $\|\cdot\|_2$ |
|---|---|---|---|---|---|---|---|
| a.4.1 | 51.9 | 53.1 | 54.4 | 56.9 | 70.0 | 73.1 | **84.4** |
| | 67.5 | 68.1 | 68.7 | 69.9 | 77.1 | 78.9 | **86.5** |
| b.1.18 | 63.8 | 62.5 | 58.1 | 65.6 | 71.9 | **75.6** | 74.4 |
| | 73.9 | 73.1 | 70.7 | 74.7 | 78.6 | **80.6** | 78.6 |
| b.29.1 | 70.6 | 80.6 | 75.6 | 77.5 | **85.0** | 83.8 | 75.0 |
| | 55.6 | 76.3 | 67.0 | 70.2 | **83.5** | 82.1 | 71.0 |
| b.40.4 | 66.9 | 68.1 | 59.4 | 70.0 | 71.9 | 68.8 | **76.2** |
| | 74.8 | 75.4 | 71.1 | 76.7 | 77.3 | 76.3 | **78.5** |
| c.1.8 | 58.8 | 75.0 | 66.9 | 73.7 | 80.6 | **85.0** | 80.0 |
| | 29.5 | 65.1 | 50.1 | 62.7 | 74.6 | **80.8** | 77.9 |
| c.3.1 | 91.9 | 97.5 | 95.6 | 95.0 | 95.6 | 95.6 | 96.2 |
| | 90.9 | 97.4 | 95.4 | 94.7 | 95.2 | 95.4 | 96.0 |
| c.47.1 | 88.1 | 86.2 | 76.2 | 90.0 | 84.4 | **90.6** | 84.4 |
| | 85.7 | 87.8 | 81.5 | 88.8 | 85.7 | **90.3** | 86.4 |
| c.67.1 | 88.8 | 90.6 | 90.6 | 91.2 | 81.9 | 91.2 | **93.1** |
| | 87.2 | 89.6 | 89.6 | 90.3 | 76.8 | 90.3 | **92.6** |

originally proposed to handle single similarity matrix, to multiple similarity matrices by averaging over the similarity matrices. We implement the linear SVM by considering similarities as feature and computing a linear kernel. We also compare with a multiple kernel learning formulation, simple MKL [12]. Denoised version of the similarity matrices are given as input to simple MKL. In Table 3 the proposed multiple similarity based kernel learning algorithms ESKL / EMKL / REKL are compared with the other methods mentioned above. We observe significant performance improvement in most cases. We also note that REKL is computationally cheaper than EMKL but provides reasonably good performance.

## 5 Conclusion

We have proposed three formulations, (4), (6), (7) for learning kernels from multiple similarity matrices. The key advantages of the proposed algorithms over the state of the art are: (i) require only SVM / MKL solvers and does not require any other sophisticated tools; (ii) the algorithms are applicable for a wide choice of loss functions and multiple similarity functions. Proposed methods can also be seen as an alternative to Multiple Kernel learning,which will be explored in future research.

## Acknowledgments

Prof. Chiranjib Bhattacharyya was partly supported by Yahoo! faculty award grant.

## Footnotes

[1] http://idl.ee.washington.edu/SimilarityLearning/

[2] http://net.icgeb.org/benchmark/

# References

[1] Stephen F. Altschul, Thomas L. Madden, Alejandro A. Schffer, Ro A. Schffer, Jinghui Zhang, Zheng Zhang, Webb Miller, and David J. Lipman. Gapped blast and psiblast: a new generation of protein database search programs. *NUCLEIC ACIDS RES*, 25:3389–3402, 1997.

[2] A. Beck and M. Teboulle. Mirror descent and nonlinear projected subgradient methods for convex optimization. *Operations Research Letters*, 31:167–175, 2003.

[3] A. Ben-Tal, T. Margalit, and A. Nemirovski. The ordered subsets mirror descent optimization method with applications to tomography. *SIAM J. Optim.*, 12:79–108, 2001.

[4] Chih-Chung Chang and Chih-Jen Lin. *LIBSVM: a library for support vector machines*, 2001. Software available at `http://www.csie.ntu.edu.tw/~cjlin/libsvm`.

[5] J. Chen and J. Ye. Training svm with indefinite kernels. In *International Conference on Machine Learning*. 2008.

[6] Y. Chen, M. R. Gupta, and B. Recht. Learning kernels from indefinite similarities. In *International Conference on Machine Learning*. 2009.

[7] G. R. Gert Lanckriet, N. Cristianini, P. Bartlett, L. E. Ghaoui, and Michael I. Jordan. Learning the kernel matrix with semidefinite programming. *Journal of Machine Learning Research*, 5:27–72, 2004.

[8] R. Luss and A. d'Aspremont. Support vector machine classification with indefinite kernels. In *Advances in Neural Information processing Systems*. 2007.

[9] A. G. Murzin, S. E. Brenner, T. Hubbard, and C. Chothia. Scop: a structural classification of proteins database for the investigation of sequences and structures. *Journal of Molecular Biology*, 247:536–540, 1995.

[10] J. Saketha Nath, G. Dinesh, S. Raman, C. Bhattacharyya, A. Ben-Tal, and K.R. Ramakrishnan. On the algorithmics and applications of a mixed-norm based kernel learning formulation. In *Advances in Neural Information Processing Systems*, pages 844–852. 2009.

[11] Saul B. Needleman and Christian D. Wunsch. A general method applicable to the search for similarities in the amino acid sequence of two proteins. *Journal of Molecular Biology*, 48(3):443–453, 1970.

[12] A. Rakotomamonjy, Francis R. Bach, S. Canu, and Y. Grandvalet. Simplemkl. *Journal of Machine Learning Research*, 9:2491–2521, 2008.

[13] Bernhard Schölkopf and Alexander J. Smola. *Learning with Kernels: Support Vector Machines, Regularization, Optimization, and Beyond (Adaptive Computation and Machine Learning)*. The MIT Press, 2001.

[14] T. F. Smith and M. S. Waterman. Identification of common molecular subsequences. *Journal of Molecular Biology*, 147(1):195 – 197, 1981.

[15] G. Wu, Z. Zhang, and E. Y. Chang. An analysis of transformation on non-positive semidefinite similarity matrix for kernel machines. *Technical Report, University of California, Santa Barbara*, 2005.

[16] Y. Ying, C. Campbell, and M. Girolami. Analysis of SVM with indefinite kernels. In *Advances in Neural Information processing Systems*, 2009.

